# Reverse TDNN: An Architecture for Trajectory Generation

**Patrice Simard**
AT&T Bell Laboratories
101 Crawford Corner Rd
Holmdel, NJ 07733

**Yann Le Cun**
AT&T Bell Laboratories
101 Crawford Corner Rd
Holmdel, NJ 07733

## Abstract

The backpropagation algorithm can be used for both recognition and generation of time trajectories. When used as a recognizer, it has been shown that the performance of a network can be greatly improved by adding structure to the architecture. The same is true in trajectory generation. In particular a new architecture corresponding to a "reversed" TDNN is proposed. Results show dramatic improvement of performance in the generation of hand-written characters. A combination of TDNN and reversed TDNN for compact encoding is also suggested.

## 1 INTRODUCTION

Trajectory generation finds interesting applications in the field of robotics, automation, filtering, or time series prediction. Neural networks, with their ability to learn from examples, have been proposed very early on for solving non-linear control problems adaptively. Several neural net architectures have been proposed for trajectory generation, most notably recurrent networks, either with discrete time and external loops (Jordan, 1986), or with continuous time (Pearlmutter, 1988). Aside from being recurrent, these networks are not specifically tailored for trajectory generation. It has been shown that specific architectures, such as the Time Delay Neural Networks (Lang and Hinton, 1988), or convolutional networks in general, are better than fully connected networks at recognizing time sequences such as speech (Waibel et al., 1989), or pen trajectories (Guyon et al., 1991). We show that special architectures can also be devised for trajectory generation, with dramatic performance improvement.

Two main ideas are presented in this paper. The first one rests on the assumption that most trajectory generation problems deal with *continuous* trajectories. Following (Pearlmutter, 1988), we present the "differential units", in which the total input to the neuron controls the em rate of change (time derivative) of that unit state, instead of directly controlling its state. As will be shown the "differential units" can be implemented in terms of regular units.

The second idea comes from the fact that trajectories are usually come from a *plan*, resulting in the execution of a "motor program". Executing a complete motor program will typically involve executing a hierarchy of sub-programs, modified by the information coming from sensors. For example drawing characters on a piece of paper involves deciding which character to draw (and what size), then drawing each stroke of the character. Each stroke involves particular sub-programs which are likely to be common to several characters (straight lines of various orientations, curved lines, loops...). Each stroke is decomposed in precise motor patterns. In short, a plan can be described in a hierarchical fashion, starting from the most abstract level (which object to draw), which changes every half second or so, to the lower level (the precise muscle activation patterns) which changes every 5 or 10 milliseconds. It seems that this scheme can be particularly well embodied by an "Oversampled Reverse TDNN". a multilayer architecture in which the states of the units in the higher layers are updated at a faster rate than the states of units in lower layers. The ORTDNN resembles a Subsampled TDNN (Bottou et al., 1990)(Guyon et al., 1991), or a subsampled weight-sharing network (Le Cun et al., 1990a), in which all the connections have been reversed, and the input and output have been interchanged. The advantage of using the ORTDNN, as opposed to a table lookup, or a memory intensive scheme, is the ability to generalize the learned trajectories to unseen inputs (plans). With this new architecture it is shown that trajectory generation problems of large complexity can be solved with relatively small resources.

## 2    THE DIFFERENTIAL UNITS

In a time continuous network, the forward propagation can be written as:

$$T\frac{\partial x(t)}{\partial t} = -x(t) + g(wx(t)) + I(t) \tag{1}$$

where $x(t)$ is the activation vector for the units, $T$ is a diagonal matrix such that $T_{ii}$ is the time constant for unit $i$, $I^t$ is the input vector at time $t$, $w$ is a weight matrix such that $w_{ij}$ is the connection from unit $j$ to unit $i$, and $g$ is a differentiable (multi-valued) function.

A reasonable discretization of this equation is:

$$\tilde{x}^{t+1} = \tilde{x}^t + \Delta t T^{-1}(-\tilde{x}^t + g(w\tilde{x}^t) + I^t) \tag{2}$$

where $\Delta t$ is the time step used in the discretization, the superscript $t$ means at time $t\Delta t$ (i.e. $\tilde{x}^t = \tilde{x}(t\Delta t)$). $x_0$ is the starting point and is a constant. $t$ ranges from 0 to $M$, with $I^0 = 0$.

The cost function to be minimized is:

$$E = \frac{1}{2} \sum_{t=1}^{t=M} (S^t \tilde{x}^t - d^t)^\top (S^t \tilde{x}^t - d^t) \tag{3}$$

Where $D^t$ is the desired output, and $S^t$ is a rectangular matrix which has a 0 if the corresponding $x_i^t$ is unconstrained and 1 otherwise. Each pattern is composed of pairs $(I^t, D^t)$ for $t \in [1..M]$. To minimize equation 3 with the constraints given by equation 2 we express the Lagrage function (Le Cun, 1988):

$$L = \frac{1}{2} \sum_{t=1}^{t=M} (S^t \tilde{x}^t - D^t)(S^t \tilde{x}^t - D^t)^T + \sum_{t=0}^{t=M-1} (\tilde{b}^{t+1})^\top (-\tilde{x}^{t+1} + \tilde{x}^t + \Delta t T^{-1}(-\tilde{x}^t + g(w\tilde{x}^t) + I^t)) \tag{4}$$

Where $\tilde{b}^{t+1}$ are Lagrange multipliers (for $t \in [1..M]$). The superscript $^T$ means that the corresponding matrix is transposed. If we differentiate with respect to $\tilde{x}^t$ we get:

$$\left(\frac{\partial L}{\partial \tilde{x}^t}\right)^T = \vec{0} = (S^t \tilde{x}^t - d^t) - \tilde{b}^t + \tilde{b}^{t+1} - \Delta t T^{-1} \tilde{b}^{t+1} - \Delta t T^{-1} w^T g'(w\tilde{x}^t) \tilde{b}^{t+1} \tag{5}$$

For $t \in [1..M-1]$ and $\frac{\partial L}{\partial \tilde{x}^M} = 0 = (S^t \tilde{x}^M - D^M) - \tilde{b}^M$ for the boundary condition. $g'$ a diagonal matrix containing the derivatives of $g$ ($g'(wx)w$ is the jacobian of $g$). From this an update rule for $\tilde{b}^t$ can be derived:

$$\begin{aligned} \tilde{b}^M &= (S^M \tilde{x}^M - d^M) \\ \tilde{b}^t &= (S^t \tilde{x}^t - d^t) + (1 - \Delta t T^{-1})\tilde{b}^{t+1} + \Delta t T^{-1} w^T \nabla g(w\tilde{x}^t) \tilde{b}^{t+1} \quad \text{for } t \in [1..M-1] \end{aligned} \tag{6}$$

This is the rule used to compute the gradient (backpropagation). If the Lagrangian is differentiated with respect to $w_{ij}$, the standard updating rule for the weight is obtained:

$$\frac{\partial L}{\partial w_{ij}} = \Delta t T^{-1} \sum_{t=1}^{t=M-1} \tilde{b}_i^{t+1} \tilde{x}_j^t g_i'(\sum_k w_{ik} \tilde{x}_k^t) \tag{7}$$

If the Lagrangian is differentiated with respect to $T$, we get:

$$\frac{\partial L}{\partial T} = -T^{-1} \sum_{t=0}^{t=M-1} (\tilde{x}^{t+1} - \tilde{x}^t) \tilde{b}^{t+1} \tag{8}$$

From the last two equations, we can derived a learning algorithm by gradient descent

$$\Delta w_{ij} = -\eta_w \frac{\partial L}{\partial w_{ij}} \tag{9}$$

$$\Delta \frac{1}{T_{ii}} = -\eta_T \frac{\partial L}{\partial \frac{1}{T_{ii}}} = -\eta_T \Delta t T_{ii} \sum_{t=0}^{t=M-1} (\tilde{x}^{t+1} - \tilde{x}^t) \tilde{b}^{t+1} \tag{10}$$

where $\eta_w$ and $\eta_T$ are respectively the learning rates for the weights and the time constants (in practice better results are obtained by having different learning rates $\eta_{w_{ij}}$ and $\eta_{T_{ii}}$ per connections). The constant $\eta_T$ must be chosen with caution

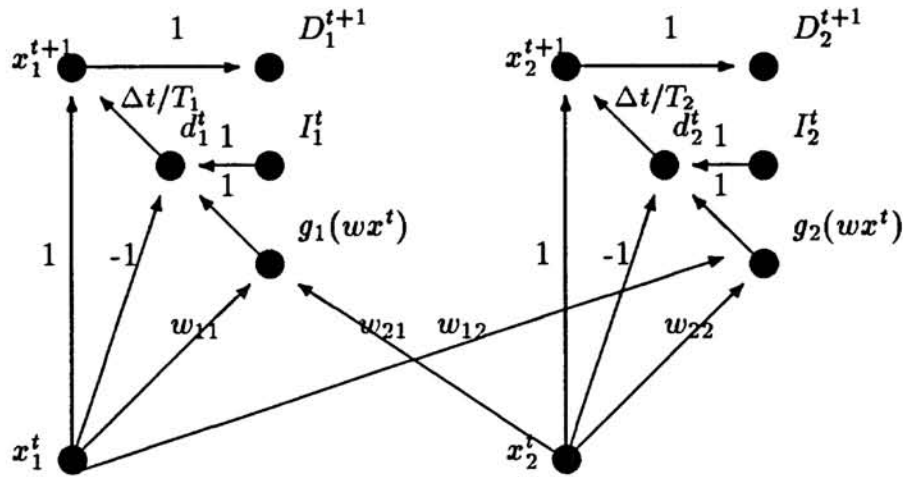

Figure 1: A backpropagation implementation of equation 2 for a two units network between time $t$ and $t + 1$. This figure repeats itself vertically for every time step from $t = 0$ to $t = M$. The quantities $x_1^{t+1}$, $x_2^{t+1}$, $d_1^t = -x_1^t + g_1(wx^t) + I_1^t$ and $d_2^t = -x_2^t + g_2(wx^t) + I_2^t$ are computed with linear units.

since if any time constants $T_{ii}$ were to become less than one, the system would be unstable. Performing gradient descent in $\frac{1}{T_{ii}}$ instead of in $T_{ii}$ is preferable for numerical stability reasons.

Equation 2 is implemented with a feed forward backpropagation network. It should first be noted that this equation can be written as a linear combination of $x^t$ (the activation at the previous time), the input, and a non-linear function $g$ of $wx^t$. Therefore, this can be implemented with two linear units and one nonlinear unit with activation function $g$. To keep the time constraint, the network is "unfolded" in time , with the weights shared from one time step to another. For instance a simple two fully connected units network with no threshold can be implemented as in Fig. 1 (only the layer between time $t$ and $t + 1$ is shown). The network repeats itself vertically for each time step with the weights shared between time steps. The main advantage of this implementation is that all equations 6, 7 and 8 are implemented implicitly by the back-propagation algorithm.

## 3   CHARACTER GENERATION: LEARNING TO GENERATE A SINGLE LETTER

In this section we describe a simple experiment designed to 1) illustrate how trajectory generation can be implemented with a recurrent network, 2) to show the advantages of using differential units instead of the traditional non linear units and 3) to show how the fully connected architecture (with differential units) severely limits the learning capacity of the network. The task is to draw the letter "A" with

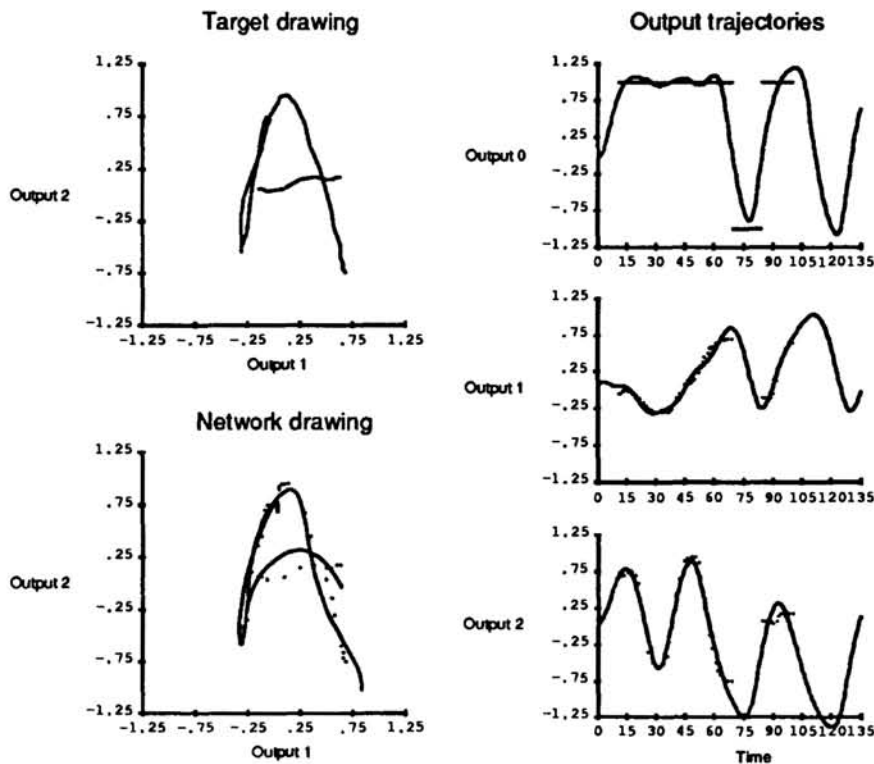

Figure 2: Top left: Trajectory representing the letter "A". Bottom left: Trajectory produced by the network after learning. The dots correspond to the target points of the original trajectory. The curve is produced by drawing output unit 2 as a function of output unit 1, using output unit 0 for deciding when the pen is up or down. Right: Trajectories of the three output units (pen-up/pen-down, X coordinate of the pen and Y coordinate of the pen) as a function of time. The dots corresponds to the target points of the original trajectory.

a pen. The network has 3 output units, two for the X and Y position of the pen, and one to code whether the pen is up or down. The network has a total 21 units, no input unit, 18 hidden units and 3 output units. The network is fully connected.

Character glyphs are obtained from a tablet which records points at successive instants of time. The data therefore is a sequence of triplets indicating the time, and the X and Y positions. When the pen is up, or if there are no constraint for some specific time steps (misreading of the tablet), the activation of the unit is left unconstrained. The letter to be learned is taken from a handwritten letter database and is displayed in figure 2 (top left).

The letter trajectory covers a maximum of 90 time stamps. The network is unfolded 135 steps (10 unconstrained steps are left at the begining to allow the network to settle and 35 additional steps are left at the end to monitor the network activity). The learning rate $\eta_w$ is set to 1.0 (the actual learning rate is per connection and is obtained by dividing the global learning rate by the fanin to the destination unit, and by dividing by the number of connections sharing the same weight). The time constants are set to 10 to produce a smooth trajectory on the output. The learning rate $\eta_T$ is equal to zero (no learning on the time constants). The initial values for the weights are picked from a uniform distribution between -1 and +1.

The trajectories fo units 0, 1 and 2 are shown in figure 2 (right). The top graphs represent the state of the pen as a function of time. The straight lines are the desired positions (1 means pen down, -1 means pen up). The middle and bottom graphs are the X and Y positions of the pen respectively. The network is unconstrained after time step 100. Even though the time constants are large, the output units reach the right values before time step 10. The top trajectory (pen-up/pen-down), however, is difficult to learn with time constants as large as 10 because it is not smooth.

The letter drawn by the network after learning is shown in figure 2 (left bottom). The network successfully learned to draw the letter on the fully connected network. Different fixed time constants were tried. For small time constant (like 1.0), the network was unable to learn the pattern for any learning rate $\eta_w$ we tried. This is not surprising since the (vertical) weight sharing makes the trajectories very sensitive to any variation of the weights. This fact emphasizes the importance of using differential units. Larger time constants allow larger learning rate for the weights. Of course, if those are too large, fast trajectories can not be learned.

The error can be further improved by letting the time constant adapt as well. However the gain in doing so is minimal. If the learning rate $\eta_T$ is small, the gain over $\eta_T = 0$ is negligible. If $\eta_T$ is too big, learning becomes quickly unstable.

This simulation was done with no input, and the target trajectories were for the drawing of a single letter. In the next section, the problem is extended to that of learning to draw multiple letters, depending on an input vector.

## 4   LEARNING TO GENERATE MULTIPLE LETTERS: THE REVERSE TDNN ARCHITECTURE

In a first attempt, the fully connected network of the previous section was used to try to generate the eight first letters of the alphabet. Eight units were used for the input, 3 for the output, and various numbers of hidden units were tried. Every time, all the units, visible and hidden, were fully interconnected. Each input unit was associated to one letter, and the input patterns consisted of one +1 at the unit corresponding to the letter, and -1/7 for all other input units. No success was achieved for all the set of parameters which were tried. The error curves reached plateaus, and the letter glyphs were not recognizable. Even bringing the number of letter to two (one "A" and one "B") was unsuccessful. In all cases the network was acting like it was ignoring its input: the activation of the output units were almost identical for all input patterns. This was attributed to the network architecture.

A new kind of architecture was then used, which we call "Oversampled Reverse TDNN" because of its resemblance with a Subsampled TDNN with input and output interchanged. Subsampled TDNN have been used in speech recognition (Bottou et al., 1990), and on-line character recognition (Guyon et al., 1991). They can be seen one-dimensional versions of locally-connected, weight sharing networks (Le Cun, 1989)(Le Cun et al., 1990b). Time delay connections allow units to be connected to unit at an earlier time. Weight sharing in time implements a convolution of the input layer. In the Subsampled TDNN, the rate at which the units states are updated decreases gradually with the layer index. The subsampling provides

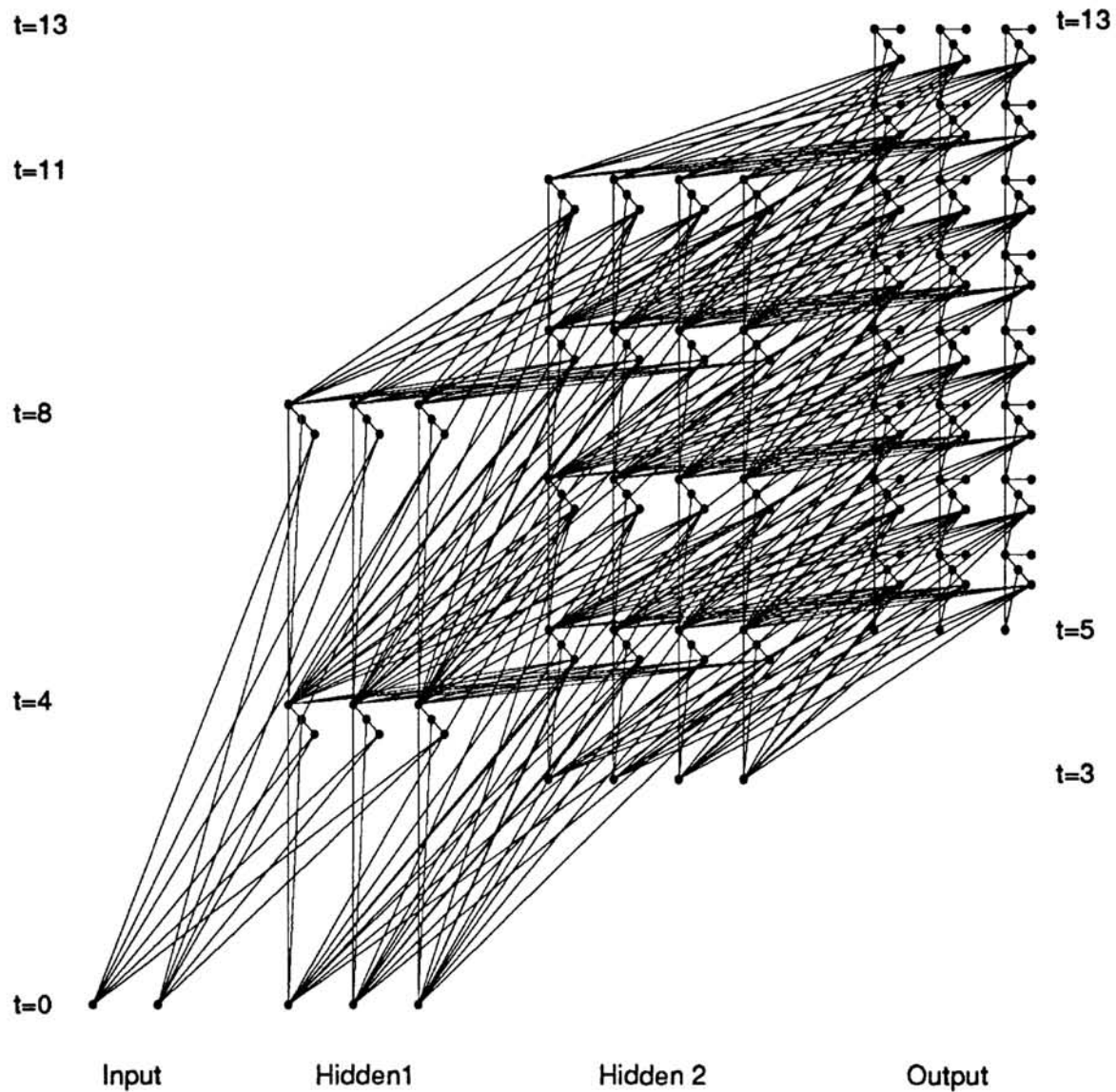

Figure 3: Architecture of a simple reverse TDNN. Time goes from bottom to top, data flows from left to right. The left module is the input and has 2 units. The next module (hidden1) has 3 units and is undersampled every 4 time steps. The following module (hidden2) has 4 units and is undersampled every 2 time steps. The right module is the output, has 3 units and is not undersampled. All modules have time delay connections from the preceding module. Thus the hidden1 is connected to hidden2 over a window of 5 time steps, and hidden2 to the output over a window of 3 time steps. For each pattern presented on the 2 input units, a trajectory of 8 time steps is produced by the network on each of the 3 units of the output.

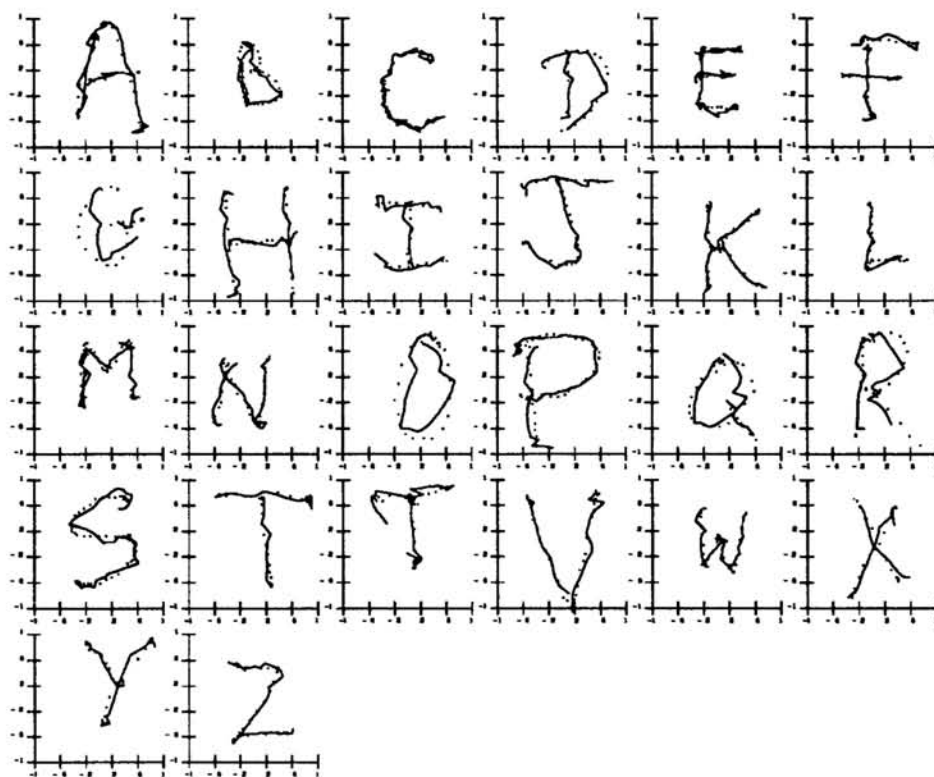

Figure 4: Letters drawn by the reverse TDNN network after 10000 iteration of learning.

a gradual reduction of the time resolution. In a reverse TDNN the subsampling starts from the units from the output (which have no subsampling) toward the input. Equivalently, each layer is oversampled when compared to the previous layer. This is illustrated in Figure 3 which shows a small reverse TDNN. The input is applied to the 2 units in the lower left. The next layer is unfolded in time two steps and has time delay connections toward step zero of the input. The next layer after this is unfolded in time 4 steps (with again time delay connections), and finally the output is completely unfolded in time. The advantage of such an architecture is its ability to generate trajectories progressively, starting with the lower frequency components at each layer. This parallels recognition TDNN's which extract features progressively. Since the weights are shared between time steps, the network on the figures has only 94 free weights.

With the reverse TDNN architecture, it was easy to learn the 26 letters of of the alphabet. We found that the learning is easier if all the weights are initialized to 0 except those with the shortest time delay. As a result, the network initially only sees its fastest connections. The influence of the remaining connections starts at zero and increase as the network learns. The glyphs drawn by the network after 10,000 training epochs are shown in figure 4. To avoid ambiguity, we give subsampling rates with respect to the output, although it would be more natural to mention oversampling rates with respect to the input. The network has 26 input units, 30 hidden units in the first layer subsampled at every 27 time steps, 25 units at the next layer subsampled at every 9 time steps, and 3 output units with no subsampling. Every layer has time delay connections from the previous layer, and is connected with 3 different updates of the previous layer. The time constants were not subject

to learning and were initialized to 10 for the x and y output units, and to 1 for the remaining units. No effort was made to optimize these values.

Big initial time constants prevent the network from making fast variations on the output units and in general slow down the learning process. On the other hand, small time constants make learning more difficult. The correct strategy is to adapt the time constants to the intrinsic frequencies of the trajectory. With all the time constants equal to one, the network was not able to learn the alphabet (as it was the case in the experiment of the previous section). Good results are obtained with time constants of 10 for the two x-y output units and time constants of 1 for all other units.

## 5   VARIATIONS OF THE ORTDNN

Many variations of the Oversampled Reverse TDNN architecture can be imagined. For example, recurrent connections can be added: connections can go from right to left on figure 3, as long as they go up. Recurrent connections become necessary when information needs to be stored for an arbitrary long time. Another variation would be to add sensor inputs at various stages of the network, to allow adjustment of the trajectory based on sensor data, either on a global scale (first layers), or locally (last layers). Tasks requiring recurrent ORTDNN's and/or sensor input include dynamic robot control or speech synthesis.

Another interesting variation is an encoder network consisting of a Subsampled TDNN and an Oversmapled Reverse TDNN connected back to back. The Subsampled TDNN encodes the time sequence shown on its input, and the ORTDNN reconstructs an time sequence from the output of the TDNN. The main application of this network would be the compact encoding of time series. This network can be trained to reproduce its input on its output (auto-encoder), in which case the state of the middle layer can be used as a compact code of the input sequence.

## 6   CONCLUSION

We have presented a new architecture capable of learning to generate trajectories efficiently. The architecture is designed to favor hierarchical representations of trajectories in terms of subtasks.

The experiment shows how the ORTDNN can produce different letters as a function of the input. Although this application does not have practical consequences, it shows the learning capabilities of the model for generating trajectories. The task presented here was particularly difficult because there is no correlation between the patterns. The inputs for an A or a Z only differ on 2 of the 26 input units. Yet, the network produces totally different trajectories on the output units. This is promising since typical neural net application have very correlated patterns which are in general much easier to learn.

## References

Bottou, L., Fogelman, F., Blanchet, P., and Liénard, J. S. (1990). Speaker inde-

pendent isolated digit recognition: Multilayer perceptron vs Dynamic Time Warping. *Neural Networks*, 3:453–465.

Guyon, I., Albrecht, P., Le Cun, Y., Denker, J. S., and W., H. (1991). design of a neural network character recognizer for a touch terminal. *Pattern Recognition*, 24(2):105–119.

Jordan, M. I. (1986). Serial Order: A Parallel Distributed Processing Approach. Technical Report ICS-8604, Institute for Cognitive Science, University of California at San Diego, La Jolla, CA.

Lang, K. J. and Hinton, G. E. (1988). A Time Delay Neural Network Architecture for Speech Recognition. Technical Report CMU-cs-88-152, Carnegie-Mellon University, Pittsburgh PA.

Le Cun, Y. (1988). A theoretical framework for Back-Propagation. In Touretzky, D., Hinton, G., and Sejnowski, T., editors, *Proceedings of the 1988 Connectionist Models Summer School*, pages 21–28, CMU, Pittsburgh, Pa. Morgan Kaufmann.

Le Cun, Y. (1989). Generalization and Network Design Strategies. In Pfeifer, R., Schreter, Z., Fogelman, F., and Steels, L., editors, *Connectionism in Perspective*, Zurich, Switzerland. Elsevier. an extended version was published as a technical report of the University of Toronto.

Le Cun, Y., Boser, B., Denker, J. S., Henderson, D., Howard, R. E., Hubbard, W., and Jackel, L. D. (1990a). Handwritten digit recognition with a back-propagation network. In Touretzky, D., editor, *Advances in Neural Information Processing Systems 2 (NIPS*89)*, Denver, CO. Morgan Kaufman.

Le Cun, Y., Boser, B., Denker, J. S., Henderson, D., Howard, R. E., Hubbard, W., and Jackel, L. D. (1990b). Back-Propagation Applied to Handwritten Zipcode Recognition. *Neural Computation*.

Pearlmutter, B. (1988). Learning State Space Trajectories in Recurrent Neural Networks. *Neural Computation*, 1(2).

Waibel, A., Hanazawa, T., Hinton, G., Shikano, K., and Lang, K. (1989). Phoneme Recognition Using Time-Delay Neural Networks. *IEEE Transactions on Acoustics, Speech and Signal Processing*, 37:328–339.
